# Computing Marginal Distributions over Continuous Markov Networks for Statistical Relational Learning

**Matthias Bröcheler,   Lise Getoor**
University of Maryland, College Park
College Park, MD 20742
{matthias, getoor}@cs.umd.edu

## Abstract

Continuous Markov random fields are a general formalism to model joint probability distributions over events with continuous outcomes. We prove that marginal computation for constrained continuous MRFs is #P-hard in general and present a polynomial-time approximation scheme under mild assumptions on the structure of the random field. Moreover, we introduce a sampling algorithm to compute marginal distributions and develop novel techniques to increase its efficiency. Continuous MRFs are a general purpose probabilistic modeling tool and we demonstrate how they can be applied to statistical relational learning. On the problem of collective classification, we evaluate our algorithm and show that the standard deviation of marginals serves as a useful measure of confidence.

## 1  Introduction

Continuous Markov random fields are a general and expressive formalism to model complex probability distributions over multiple continuous random variables. Potential functions, which map the values of sets (cliques) of random variables to real numbers, capture the dependencies between variables and induce a exponential family density function as follows: Given a finite set of $n$ random variables $\mathbf{X} = \{X_1, \ldots, X_n\}$ with an associated bounded interval domain $D_i \subset \mathbb{R}$, let $\phi = \{\phi_1, \ldots, \phi_m\}$ be a finite set of $m$ *continuous* potential functions defined over the interval domains, i.e. $\phi_j : \mathbf{D} \to [0, M]$, for some bound $M \in \mathbb{R}^+$, where $\mathbf{D} = D_1 \times D_2 \ldots \times D_n$. For a set of free parameters $\Lambda = \{\lambda_1, \ldots, \lambda_m\}$, we then define the probability measure $\mathbb{P}$ over $\mathbf{X}$ with respect to $\phi$ through its density function $f$ as:

$$f(\mathbf{x}) = \frac{1}{Z(\Lambda)}\exp[-\sum_{j=1}^{m}\lambda_j\phi_j(\mathbf{x})] \;\; ; \;\; Z(\Lambda) = \int_{\mathbf{D}} \exp\left(-\sum_{j=1}^{m}\lambda_j\phi_j(\mathbf{x})\right) d\mathbf{x} \qquad (1)$$

where $Z$ is the normalization constant. The definition is analogous to the popular discrete Markov random fields (MRF) but using integration over the bounded domain rather than summation for the partition function $Z$.

In addition, we assume the existence of a set of $k_A$ equality and $k_B$ inequality constraints on the random variables, that is, $A(\mathbf{x}) = a$, where $A : \mathbf{D} \to \mathbb{R}^{k_A}, a \in \mathbb{R}^{k_A}$ and $B(\mathbf{x}) \leq b$, where $B : \mathbf{D} \to \mathbb{R}^{k_B}, b \in \mathbb{R}^{k_B}$. Both equality and inequality constraints restrict the possible combinations of values the random variables $\mathbf{X}$ can assume. That is, we set $f(\mathbf{x}) = 0$ whenever any of the constraints are violated and constrain the domain of integration, denoted $\tilde{\mathbf{D}}$, for the normalization constant correspondingly. Constraints are useful in probabilistic modeling to exclude inconsistent outcomes based on prior knowledge about the distribution. We call this class of MRFs *constrained continuous Markov random fields* (CCMRF).

Probabilistic inference often requires the computation of marginal distributions for all or a subset of the random variables $\mathbf{X}$. Marginal computation for discrete MRFs has been extensively studied due to its wide applicability in probabilistic reasoning. In this work, we study the theoretical and practical aspects of computing marginal density functions over CCMRFs. General continuous MRFs can

| |
|---|
| $\lambda_1 : A.text \cong B.text \ \tilde{\wedge} \ class(A, C) \ \tilde{\Rightarrow} \ class(B, C)$ |
| $\lambda_2 : link(A, B) \ \tilde{\wedge} \ class(A, C) \ \tilde{\Rightarrow} \ class(B, C)$ |
| Constraint : $functional(class)$ |

Table 1: Example PSL program for collective classification.

be used in a variety of probabilistic modeling scenarios and have been studied for applications with continuous domains such as computer vision. Gaussian Random Fields are a type of continuous MRF which assume normality. In this work, we make no restrictive assumptions about the marginal distributions other than boundedness. For general continuous MRFs, non-parametric belief propagation (NBP) [1] has been proposed as a method to estimate marginals. NBP represents the "belief" as a combination of kernel densities which are propagated according to the structure of the MRF. In contrast to NBP, our approach provides polynomial-time approximation guarantees and avoids the representational choice of kernel densities.

The main contributions of this work are described in Section 3. We begin by showing that computing marginals in CCMRFs is #P-hard in the number of random variables $n$. We then discuss a Markov chain Monte Carlo (MCMC) sampling scheme that can approximate the exact distribution to within $\epsilon$ error in polynomial time under the general assumption that the potential functions and inequality constraints are convex. Based on this result, we propose a tractable sampling algorithm and present a novel approach to increasing its effectiveness by detecting and counteracting slow convergence. Our theoretical results are based on recent advances in computational geometry and the study of log-concave functions [2]. In Section 4, we investigate the performance, scalability, and convergence of the sampling algorithm on the probabilistic inference problem of collective classification on a set of Wikipedia documents. In particular, we show that the standard deviation of the marginal density function can serve as a strong indicator for the "confidence" in the classification prediction, thereby demonstrating a useful qualitative aspect of marginals over continuous MRFs. Before turning to the main contributions of the paper, in the next section, we give background motivation for the form of CCMRFs we study.

## 2 Motivation

Our treatment of CCMRFs is motivated by probabilistic similarity logic (PSL) [3]. PSL is a relational language that provides support for probabilistic reasoning about similarities. PSL is similar to existing SRL models, e.g., MLNs [4], BLPs [5], RMNs [6], in that it defines a probabilistic graphical model over the properties and relations of the entities in a domain as a grounding of a set of rules that have attached parameters. However, PSL supports reasoning about "soft" truth values, which can be seen as similarities between entities or sets of entities, degrees of belief, or strength of relationships. PSL uses annotated logic rules to capture the dependency structure of the domain, based on which it builds a joint continuous probabilistic model over all decision atoms which can be expressed as a CCMRF as defined above. PSL has been used to reason about the similarity between concepts from different ontologies as well as articles from Wikipedia. Table 1 shows a simple PSL program for collective classification. The first rule states that documents with similar text are likely to have the same class. The second rule says that two documents which are linked to each other are also likely to be assigned the same class. Finally, we express the constraint that each document can have at most one class, that is, the $class$ predicate is functional and can only map to one value. Such domain specific constraints motivate our introduction of equality and inequality constraints for CCMRFs. Rules and constraints are written in first order logic formalism and are grounded out against the observed data such that each ground rule constitutes one potential function or constraint computing the truth value of the formula. Rules have an associated weight $\lambda_i$ which is used as the parameter for each associated potential function. The weights can be learned from training data.

In the following, we make some assumptions about the nature of the constraints and the potential functions motivated by the requirements of the PSL framework and the types of CCMRFs modeled therein. Firstly, we assume all domains are in the $[0, 1]$ interval which corresponds to the domain of similarity truth values in PSL. Secondly, all constraints are assumed to be linear. Thirdly, the potential functions $\phi_j$ are of the form $\phi_j(\mathbf{x}) = \max(0, o_j \cdot \mathbf{x} + q_j)$ where $o_j^T \in \mathbb{R}^n$ is a $n$-dimensional row vector and $q_j \in \mathbb{R}$. The particular form of the potential functions is motivated by the way similarity truth values are combined in PSL using t-norms (see [3] for details).

While the techniques presented in this work are not specific to PSL, a brief outline of the PSL framework helps in understanding the assumptions about the CCMRFs of interest made in our algorithm and experiments. In Section 3.5 we show how our assumptions can be relaxed while maintaining polynomial-time guarantees for applications outside the PSL framework.

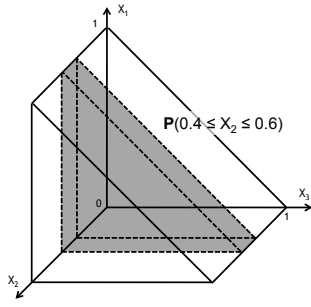 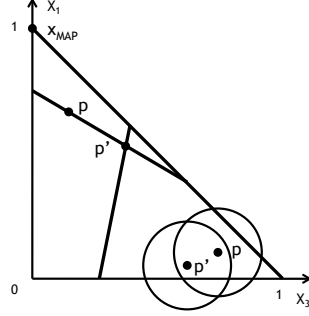

a) Example of geometric marginal computation     b) Hit-and-Run and random ball walk illustration

## 3 Computing continuous marginals

This section contains the main technical contributions of this paper. We start our study of marginal computation for CCMRFs by proving that computing the exact density function is #P hard (3.1). In Section 3.2, we discuss how to approximate the marginal distribution using a MCMC sampling scheme which produces a guaranteed $\epsilon$-approximation in polynomial time under suitable conditions. We show how to improve the sampling scheme by detecting phases of slow convergence and present a technique to counteract them (3.3). Finally, we describe an algorithm based on the sampling scheme and its improvements (3.4). In addition, we discuss how to relax the linearity conditions in Section 3.5.

Throughout this discussion we use the following simple example for illustration:

**Example 1** *Let $\mathbf{X} = \{X_1, X_2, X_3\}$ be subject to the inequality constraint $x_1 + x_3 \leq 1$. Let $\phi_1(\mathbf{x}) = x_1$, $\phi_2(\mathbf{x}) = max(0, x_1 - x_2)$, $\phi_3(\mathbf{x}) = max(0, x_2 - x_3)$ where $\lambda = (1, 2, 1)$ are the associated free parameters.*

### 3.1 Exact marginal computation

**Theorem 1** *Computing the marginal probability density function $f_{\mathbf{X}'}(\mathbf{x}') = \int_{\mathbf{y} \in \times \tilde{D}_i, s.t. X_i \notin \mathbf{X}'} f(\mathbf{x}', \mathbf{y}) d\mathbf{y}$ for a subset $\mathbf{X}' \subset \mathbf{X}$ under a probability measure $\mathbb{P}$ defined by a CCMRF is #P hard in the worst case.*

We prove this statement by a simple reduction from the problem of computing the volume of a $n$-dimensional polytope defined by linear inequality constraints. To see the relationship to computational geometry, note that the domain $\mathbf{D}$ is a $n$-dimensional unit hypercube[1]. Each linear inequality constraints $B_i$ from the system $B$ can be represented by a hyperplane which "cuts off" part of the hypercube $\mathbf{D}$. Finally, the potential functions induce a probability distribution over the resulting convex polytope. Figure 3a) visualizes the domain for our running example in the 3-dimensional Euclidean space. The constraint domain is shown as a wedge. The highlighted area marks the region of probability mass that is equal to the probability $\mathbb{P}(0.4 \leq X_2 \leq 0.6)$.

**Proof 1 (Sketch)** *For any random variable $X \in \mathbf{X}$, the marginal probability $\mathbb{P}(l \leq X \leq u)$ under the uniform probability distribution defined by a single potential function $\phi = 0$ corresponds to the volume of the "slice" defined by the bounds $u < l \in [0, 1]$ relative to the volume of the entire polytope. In [7] it was shown that computing the volume of such slices is at least as hard as computing the volume of the entire polytope which is known to be #P-hard [8].*

### 3.2 Approximate marginal computation and sampling scheme

Despite this hardness result, efficient approximation algorithms for convex volume computation based on MCMC techniques have been devised and yield polynomial-time approximation guarantees. We will review the techniques and then relate them to our problem of marginal computation.

The first provably polynomial-time approximation algorithm for volume computation was based on "random ball-walks". Starting from some initial point $p$ inside the polytope, one samples from the local density function of $f$ restricted to the inside of a ball of radius $r$ around the point $p$. If the newly sampled point $p'$ lies inside the polytope, we move to $p'$, otherwise we stay at $p$ and repeat the sampling. If $\mathbb{P}$ is the uniform distribution (as typically chosen for volume computation), the

resulting Markov chain converges to $\mathbb{P}$ over the polytope in $O^*(n^3)$ steps assuming that the starting distribution is not "too far" from $\mathbb{P}$ [9].[2]

More recently, the hit and run sampling scheme [10] was rediscovered which has the advantage that no strong assumptions about the initial distribution need to be made. As in the random ball walk, we start at some interior point $p$. Next, we generate a direction $d$ (i.e., $n$ dimensional vector of length 1) uniformly at random and compute the line segment $l$ of the line $p + \alpha d$ that resides inside the polytope. We then compute the distribution of $\mathbb{P}$ over the segment $l$, sample from it uniformly at random and move to the new sample point $p'$ to repeat the process. For $\mathbb{P}$ being the uniform distribution, the Markov chain also converges after $O^*(n^3)$ steps but for hit-and-run we only need to assume that the starting point $p$ does not lie on the boundary of the polytope [2]. In [7], the authors show that hit-and-run significantly outperforms random ball walk sampling in practice, because it (1) does not get easily stuck in corners since each sample is guaranteed to be drawn from inside the polytope, (2) does not require parameter setting like the radius $r$ which greatly influences the performance of random ball walk. Figure 3 b) shows an iteration of the random ball walk and the hit-and-run sampling schemes for our running example restricted to just two dimensions to simplify the presentation. We can see that, depending on the radius of the ball, a significant portion may not intersect with the feasible region.

Lovász and Vempala[2] have proven a stronger result which shows that hit-and-run sampling converges for general log-linear distributions. Based on their result, we get a polynomial-time approximation guarantee for distributions induced by CCMRFs as defined above.

**Theorem 2** *The complexity of computing an approximate distribution $\sigma^*$ using the hit-and-run sampling scheme such that the total variation distance of $\sigma^*$ and $\mathbb{P}$ is less than $\epsilon$ is $O^*\left(\tilde{n}^3(k_B + \tilde{n} + m)\right)$, where $\tilde{n} = n - k_A$, under the assumptions that we start from an initial distribution $\sigma$ such that the density function $d\sigma/d\mathbb{P}$ is bounded by $M$ except on a set $S$ with $\sigma(S) \le \epsilon/2$.*

**Proof 2 (Sketch)** *Since $A, B$ are linear, $\tilde{\mathbf{D}}$ is an $\tilde{n} = n - k_A$ dimensional convex polytope after dimensionality reduction through $A$. By definition, $f$ is from the exponential family and since all factors are linear or maximums of linear functions, $f$ is a log concave function (maximums and sums of convex functions are convex). More specifically, $f$ is a log concave and log piecewise linear function. Let $\sigma^s$ be the distribution of the current point after $s$ steps of hit-and-run have been applied to $f$ starting from $\sigma$. Now, according to Theorem 1.3 from [2], for $s > 10^{30}\frac{n^2 R^2}{r^2}ln^5\frac{MnR}{r\epsilon}$ the total variation distance of $\sigma^s$ and $\mathbb{P}$ is less than $\epsilon$, where $r$ is such that the level set of $f$ of probability $\frac{1}{8}$ contains a ball of radius $r$ and $R^2 \ge \mathbf{E}_f(|x - z_f|^2)$, where $z_f$ is the centroid of $f$.*
*Now, each hit-and-run step requires us to iterate over the random variable domain boundaries, $O(\tilde{n})$, compute intersections with the inequality constraints, $O(\tilde{n}k_B)$, and integrate over the line segment involving all factors, $O(\tilde{n}m)$.*

### 3.3 Improved sampling scheme

Our proposed sampling algorithm is an implementation of the hit-and-run MCMC scheme. However, the theoretical treatment presented above leaves two questions unaddressed: 1) How do we get the initial distribution $\sigma$? 2) The hit-and-run algorithm assumes that all sample points are strictly inside the polytope and bounded away from its boundary. How can we get out of corners if we do get stuck?

The theorem above assumes a suitable initial distribution $\sigma$, however, in practice, no such distribution is given. Lovász and Vempala also show that the hit-and-run scheme converges from a *single* starting point on uniform distributions under the condition that it does not lie on the boundary and at the expense of an additional factor of $n$ in the number of steps to be taken (compare Theorem 1.1 and Corollary 1.2 in [2]). We follow this approach and use a MAP state $\mathbf{x}_{MAP}$ of the distribution $\mathbb{P}$ as the single starting point for the sampling algorithm. Choosing a MAP state as the starting point has two advantages: 1) we are guaranteed that $\mathbf{x}_{MAP}$ is an interior point and 2) it is the point with the highest probability density and therefore highest probability mass in a small local neighborhood.

However, starting from a MAP state elevates the importance of the second question, since the MAP state often lies exactly on the boundary of the polytope and therefore we are likely to start the sampling algorithm from a vertex of the polytope. The problem with corner points $p$ is that most of

the directions sampled uniformly at random will lead to line segments of zero length and hence we do not move between iterations. Let $W$ be the subset of inequality constraints $B$ that are "active" at the corner point $p$ and $b$ the corresponding entries in $b$, i.e. $Wp = b$ (since all constraints are linear, we abuse notation and consider $B, W$ to be matrices). In other words, the hyperplanes corresponding to the constraints in $W$ intersect in $p$. Now, for all directions $d \in \mathbb{R}^n$ such that there exist active constraints $W_i, W_j$ with $W_i d < 0$ and $W_j d > 0$, the line segment through $p$ induced by $d$ must necessarily be $0$. It also follows that more active constraints increase the likelihood of getting stuck in a corner.

For example, in Figure 3 b) the point $\mathbf{x}_{MAP}$ in the upper left hand corner denotes the MAP state of the distribution defined in our running example. If we generate a direction uniformly at random, only $1/4$ of those will be feasible, that is, for all other we won't be able to move away from $\mathbf{x}_{MAP}$.

To avoid the problem of repeatedly sampling infeasible directions at corner points, we propose to restrict the sampling of directions to feasible directions only when we determine that a corner point has been reached. We define a corner point $p$ as a point inside the polytope where the number of active constraints is above some threshold $\theta$.[3] A direction $d$ is feasible, if $Wd < 0$. Assuming that there are $a$ active constraints at corner point $p$ (i.e., $W$ has $a$ rows) we sample each entry of the $a$-dimensional vector $z$ from $-|N(0,1)|$ where $N(0,1)$ is the standard Gaussian distribution with zero mean and unit variance. Now, we try to find directions $d$ such that $Wd \leq z$.

A number of algorithms have been proposed to solve such systems of linear inequalities for feasible points $d$. In our sampling algorithm we implement the relaxation method introduced by Agmon [11] and Motzkin and Schoenberg [12] due to its simplicity. The relaxation method proceeds as follows: We start with $d_0 = \mathbf{0}$. At each iteration we check if $Wd_i \leq z$ ; if so, we have found a solution and terminate. If not, we choose the most "violated" inequality constraint $W_k$ from $W$, i.e., the row vector $W_k$ from $W$ which maximizes $\frac{W_k d_i - z_k}{\|W_k\|}$, and update the direction,

$$d_{i+1} = d_i + 2\frac{z_k - W_k d_i}{\|W_k\|^2}W_k^T$$

The relaxation method is guaranteed to terminate, since a feasible direction $d$ always exists [12].

## 3.4 Sampling algorithm

| | **Algorithm** CCMRF Sampling |
|---|---|
| | **Input:** CCMRF specified by RVs $\mathbf{X}$ with domains $\mathbf{D} = [0,1]^n$, equality constraints |
| | $\quad A(\mathbf{x}) = a$ inequality constraints $B(\mathbf{x}) \leq b$, potential functions $\phi$, parameters $\Lambda$ |
| | **Output:** Marginal probability density histograms $H[\mathbf{X}_i]: [0,1] \to \mathbb{R}^+, \forall X_i \in \mathbf{X}$ |

| | | | |
|---|---|---|---|
| 1 | **if** $A = \emptyset$ | 27 | $\quad\quad$ **for** $i = 1 \ldots |\text{rows}(B)|$ |
| 2 | $\quad P \leftarrow \mathbf{1}_{|\mathbf{x}|}$ | 28 | $\quad\quad\quad$ **if** $cd_i \neq 0$ |
| 3 | $\quad n' \leftarrow n$ | 29 | $\quad\quad\quad\quad a = \frac{b_i - cx_i}{cd_i}$ |
| 4 | **else** | 30 | $\quad\quad\quad\quad$ **if** $cd_i > 0$ **then** $\alpha_{high} \leftarrow \min(\alpha_{high}, a)$ |
| 5 | $\quad r \leftarrow \text{rank}(A)$ | 31 | $\quad\quad\quad\quad$ **if** $cd_i < 0$ **then** $\alpha_{low} \leftarrow \max(\alpha_{low}, a)$ |
| 6 | $\quad [U, \Sigma, V] \leftarrow \text{svd}(A)$ | 32 | $\quad\quad\quad\quad$ **if** $a = 0$ **then** $active \leftarrow active \cup \{i\}$ |
| 7 | $\quad P \leftarrow V|_{\text{columns}: [r+1, n]}$ | 33 | $\quad\quad$ **if** $\alpha_{high} - \alpha_{low} = 0 \wedge |active| > \theta$ |
| 8 | $\quad n' \leftarrow n - r$ | 34 | $\quad\quad\quad cornered \leftarrow \text{TRUE}$ |
| 9 | $\quad \mathbf{x}^0 \leftarrow \text{MAP}(A(\mathbf{x}) = a, B(\mathbf{x}) \leq b, \phi)$ | 35 | $\quad\quad\quad$ **continue** |
| | | 36 | $\quad\quad M \leftarrow \text{map}: [0,1] \to \mathbb{R} \times \mathbb{R}$ |
| 10 | $cornered \leftarrow \text{FALSE}$ | 37 | $\quad\quad$ **for** $\phi_i = \max(0, o_i \cdot \mathbf{x} + q_i) \in \phi$ |
| 11 | **for** $j = 0$ to $\rho$ | 38 | $\quad\quad\quad r \leftarrow \lambda_i (o_i \cdot \mathbf{d})$ |
| 12 | $\quad$ **if** $cornered$ | 39 | $\quad\quad\quad c \leftarrow \lambda_i (o_i \cdot \mathbf{x_j} + q_i)$ |
| 13 | $\quad\quad \mathbf{d} \leftarrow \vec{0}$ | 40 | $\quad\quad\quad a \leftarrow -c/r$ |
| 14 | $\quad\quad W \leftarrow B|_{\text{rows}: active} \times P$ | 41 | $\quad\quad\quad$ **if** $r > 0 \wedge a < \alpha_{high}$ |
| 15 | $\quad\quad \mathbf{z} \leftarrow z_i = \sim -|N(0,1)| \, \forall i = 1 \ldots n'$ | 42 | $\quad\quad\quad\quad M(\max(a, \alpha_{low})) \leftarrow M(\max(a, \alpha_{low})) + [r, c]$ |
| 16 | $\quad\quad$ **while** $\exists i : W_i \mathbf{d} - z_k > 0$ | 43 | $\quad\quad\quad$ **else if** $r < 0 \wedge a > \alpha_{low}$ |
| 17 | $\quad\quad\quad v \leftarrow \text{argmax}_k \frac{W_k \mathbf{d} - z_k}{\|W_k\|}$ | 44 | $\quad\quad\quad\quad M(\alpha_{low}) \leftarrow M(\alpha_{low}) + [r, c]$ |
| 18 | $\quad\quad\quad \mathbf{d} = \mathbf{d} + 2\frac{z_v - W_v \mathbf{d}}{\|W_v\|^2}W_v^T$ | 45 | $\quad\quad\quad\quad$ **if** $a < \alpha_{high}$ **then** $M(a) \leftarrow M(a) + [-r, -c]$ |
| 19 | $\quad\quad cornered \leftarrow \text{FALSE}$ | 46 | $\quad\quad\quad$ **else** $M(\alpha_{low}) \leftarrow M(\alpha_{low}) + [0, c]$ |
| 20 | $\quad$ **else** | 47 | $\quad\quad [r_\alpha, c_\alpha] \leftarrow \sum_{a \leq \alpha} M(a)$ |
| 21 | $\quad\quad \mathbf{d} \leftarrow d_i = \sim N(0,1) \, \forall i = 1 \ldots n'$ | 48 | $\quad\quad \Sigma_\alpha \leftarrow \sum_{a < b < \alpha \wedge \nexists c: a < c < b} \frac{1}{r_a}e^{-c_a}(e^{-r_a a} - e^{-r_a b})$ |
| 22 | $\quad \mathbf{d} \leftarrow \frac{1}{\|\mathbf{d}\|}\mathbf{d}$ | 49 | $\quad\quad s \leftarrow \sim [0, \Sigma_{\alpha_{high}}]$ |
| 23 | $\quad \mathbf{d} \leftarrow P \times \mathbf{d}$ | 50 | $\quad\quad a \leftarrow \max\{\alpha \in M \mid \Sigma_\alpha \leq s\}$ |
| 24 | $\quad active \leftarrow \emptyset$ | 51 | $\quad\quad \alpha \leftarrow \frac{-1}{r_a}\left(\log\left(-sr_a + r_a\Sigma_a + e^{-c_a - r_a a}\right) + c_a\right)$ |
| 25 | $\quad \alpha_{low} \leftarrow -\infty, \alpha_{high} \leftarrow \infty$ | 52 | $\quad\quad \mathbf{x}^{j+1} \leftarrow \mathbf{x}^j + \alpha\mathbf{d}$ |
| 26 | $\quad \mathbf{cd} \leftarrow B \times \mathbf{d} \,;\, \mathbf{cx} \leftarrow B \times \mathbf{x}^j$ | 53 | $\quad\quad$ **if** $j > \frac{\rho}{100}n'^3$ |
| | | 54 | $\quad\quad\quad H[i][\mathbf{x}_i^{j+1}] \leftarrow H[i][\mathbf{x}_i^{j+1}] + 1 \, \forall i = 1 \ldots n$ |

Figure 1: Constrained continuous MRF sampling algorithm

Putting the pieces together, we present the marginal distribution sampling algorithm in Figure 1. The inputs to the algorithm were discussed in Section 1. In addition, we assume that the domain restrictions $D_i = [l, u]$ for the random variables $X_i$ are encoded as pairs of linear inequality constraints $l \leq \mathbf{x}_i \leq u$ in $B, b$. The algorithm first analyzes the equality constraints $A$ to determine the number of "free" random variables and reduce the dimensionality accordingly. The singular-value decomposition of $A$ is used to determine the $n \times n'$ projection matrix $P$ which maps from the null-space of $A$ to the original space $\mathbf{D}$, where $n' = n - rank(A)$ is the dimensionality of the null-space. If no equality constraints have been specified, $P$ is the $n$-dimensional unit matrix. Next, the algorithm determines a MAP state $\mathbf{x}^0$ of the density function defined by the CCMRF, which is the point with the highest probability mass, that is, $\mathbf{x}^0 = \text{argmax}_{x \in \tilde{\mathbf{D}}} f(x)$. Since the $Z(\Lambda)$ is constant and the logarithm monotonic, this is identical to $\mathbf{x}^0 = \text{argmin}_{x \in \tilde{\mathbf{D}}} \sum_{j=1}^{m} \lambda_i \phi_i(x)$. Hence, computing a MAP state can be cast as a linear optimization problem, since all constraints are linear and the potential functions maximums of two linear functions. Linear optimization problems can be solved efficiently in time $O(n^{3.5})$ and are very fast in practice.

After determining the null-space and starting point, we begin collecting $\rho$ samples. If we detected being stuck in a corner during the previous iteration, we sample a direction $\mathbf{d}$ from the feasible subspace of all possible directions in the reduced null-space using the adapted relaxation method described above (lines 13-19). Otherwise, we sample a direction uniformly at random from the null-space of $A$. We then normalize the direction and project it back into our original domain $\mathbf{D}$ by matrix multiplication with $P$. The projection ensures that all equality constraints remain satisfied as we move along the direction $\mathbf{d}$. Next, we compute the segment of the line $l : x^j + \alpha \mathbf{d}$ inside the polytope defined by the inequality constraints $B$ (lines 25-32). Iterating over all inequality constraints, we determine the value of $\alpha$ where $l$ intersects the constraint $i$. We keep track of the largest negative and smallest positive values to define the bounds $[\alpha_{low}, \alpha_{high}]$ such that the line segment is defined exactly by those values of $\alpha$ inside this interval. In addition, we determine all active constraints, i.e. those constraints where the current sample point $x^j$ is the point of intersection and hence $\alpha = 0$. If the interval $[\alpha_{low}, \alpha_{high}]$ is 0, then we are currently sitting in a corner. If, in addition, the number of active constraints exceed some threshold $\theta$ we are stuck in a corner and abort the current iteration to start over with restricted direction sampling.

In lines 36-48 we compute the cumulative density function of the probability $\mathbb{P}$ over the line segment $l$ with $\alpha \in [\alpha_{low}, \alpha_{high}]$. Based on our assumption in Section 2, the sum of potential functions $S = \sum_{i=1}^{m} \lambda_i \phi_i$ restricted to the line $l$ is a continuous piece-wise linear function. In order to integrate the density function, we need to segment $S$ into its differentiable parts, so we start by determining the subintervals of $[\alpha_{low}, \alpha_{high}]$ where $S$ is linear and differentiable and can therefore be described by $S = rx + c$. We compute the slope $r$ and y-intercept $c$ for each potential function individually as well as the point of undifferentiability $a$ where the line crosses 0. We use a map $M$ to store the line description $[r, c]$ with the point of intersection $a$ (lines 36-46). Then, we compute the aggregate slope $r_a$ and y-intercept $c_a$ for the sum of all potentials for each point of undifferentiability $a$ (line 47) and use this information to compute the unnormalized cumulative density function by integrating over each subinterval and summing those up in $\Sigma_\alpha$ (line 48). Now, $\Sigma_a / \Sigma_{\alpha_{high}}$ gives the cumulative probability mass for all points of undifferentiability $a$ which define the subintervals. Next, we sample a number $s$ from the interval $[0, \Sigma_{\alpha_{high}}]$ uniformly at random (line 49) and compute $\alpha$ such that $\Sigma_\alpha = s$ (line 50-51). Finally, we move to the new sample point $\mathbf{x}^{j+1} = \mathbf{x}^j + \alpha \mathbf{d}$ and add it to the histogram which approximates the marginal densities if the number of steps taken so far exceeds the burn-in period which we configured to be 1% of the total number of steps.

## 3.5 Generalizing to convex continuous MRFs

In our treatment so far, we made specific assumptions about the constraints and potential functions. More generally, Theorem 2 holds when the inequality constraints as well as the potential functions are convex. A system of inequality constraints is convex if the set of all points that satisfy the constraints is convex, that is, any line connecting two points in the set is completely contained in the set.

Our algorithm needs to be modified where we currently assume linearity. Firstly, computing a MAP state requires general convex optimization. Secondly, our method for finding feasible directions when being caught in a corner of the polytope needs to be adapted to the case of arbitrary convex constraints. One simple approach is to use the tangent hyperplane at the point $\mathbf{x}_j$ as an approximation to the actual constraint and proceed as is. Similarly, we need to modify the computation of intersection points between the line and the convex constraints as well as how we determine the

points of undifferentiability. Lastly, the computation of integrals over subintervals for the potential functions requires knowledge of the form of potential functions to be solved analytically or they need to be approximated efficiently. The algorithm can handle arbitrary domains for the random variables as long as they are connected subintervals of $\mathbb{R}$.

## 4 Experiments

This section presents an empirical evaluation of the proposed sampling algorithm on the problem of category prediction for Wikipedia documents based on similarity. After describing the data and the experimental methodology, we demonstrate that the computed marginal distributions effectively predict document categories. Moreover, we show that analysis of the marginal distribution provides an indicator for the confidence in those predictions. Finally, we investigate the convergence rate and runtime performance of the algorithm in detail.

For our evaluation dataset, we collected all Wikipedia articles that appeared in the featured list[4] for a two week period in Oct. 2009, thus obtaining 2460 documents. Of these, we considered a subset of 1717 documents assigned to the 7 most popular categories. After stemming and stop-word removal, we represented the text of each document as a tf/idf-weighted word vector. To measure the similarity between documents, we used the popular cosine metric on the weighted word vectors. The data contains the relations `Link(fromDoc, toDoc)`, which establishes a hyperlink between two documents. We used $K$-fold cross-validation for $k = 20, 25, 30, 35$ by splitting the dataset into $K$ non-overlapping subsets each of which is determined using snowball sampling over the link structure from a randomly chosen initial document. For each training and test data subset, we randomly designate 20% of the documents as "seed documents" of which the category is observed and the goal is to predict the categories of the remaining documents. All experiments were executed on identical hardware powered by two Intel Xeon Quad Core 2.3 GHz Processors and 8 GB of RAM.

### 4.1 Classification results

| $K$ | Baseline | Marginals | Improvement |
|-----|----------|-----------|-------------|
| 20  | 39.5%    | 55.8%     | 41.4%       |
| 25  | 39.1%    | 51.5%     | 31.7%       |
| 30  | 36.7%    | 51.1%     | 39.1%       |
| 35  | 38.8%    | 56.6%     | 46.1%       |

| $K$ | $\mathbb{P}$(Null Hypothesis) | Relative Difference $\Delta(\sigma)$ |
|-----|-------------------------------|--------------------------------------|
| 20  | 1.95E-09                      | 38.3%                                |
| 25  | 2.40E-13                      | 41.2%                                |
| 30  | <1.00E-16                     | 43.5%                                |
| 35  | 4.54E-08                      | 39.0%                                |

Figure 2: a) Classification Accuracy    b) Std. deviation as an indicator for confidence

The baseline method uses only the document content by propagating document categories via textual similarity measured by the cosine distance. Using rules and constraints similar to those presented in Table 1, we create a joint probabilistic model for collective classification of Wikipedia documents. We use PSL twofold in this process: Firstly, PSL constructs the CCMRF by grounding the rules and constraints against the given data as described in Section 2 and secondly, we use the perceptron weight learning method provided by PSL to learn the free parameters of the CCMRF from the training data (see [3] for more detail). The sampling algorithm takes the constructed CCMRF and learned parameters as input and computes the marginal distributions for all random variables from 3 million samples. We have one random variable to represent the similarity for each possible document-category pair, that is, one RV for each grounding of the `category` predicate. For each document $D$ we pick the category $C$ with the highest expected similarity as our prediction. The accuracy in prediction of both methods is compared in Table 2 a) over the 4 different splits of the data. We observe that the collective probabilistic model outperforms the baseline by up to 46%. All results are statistically significant at $p = 0.02$.

While this results suggests that the sampling algorithm works in practice, it is not surprising and novel since similar results for collective classification have been produced before using other approaches in statistical relational learning (e.g., compare [13]). However, the marginal distributions we obtain provide additional information beyond the simple point estimate of its expected value. In particular, we show that the standard deviation of the marginals can serve as an indicator for the confidence in the particular classification prediction. In order to show this, we compute the standard deviation of the marginal distributions for those random variables picked during the

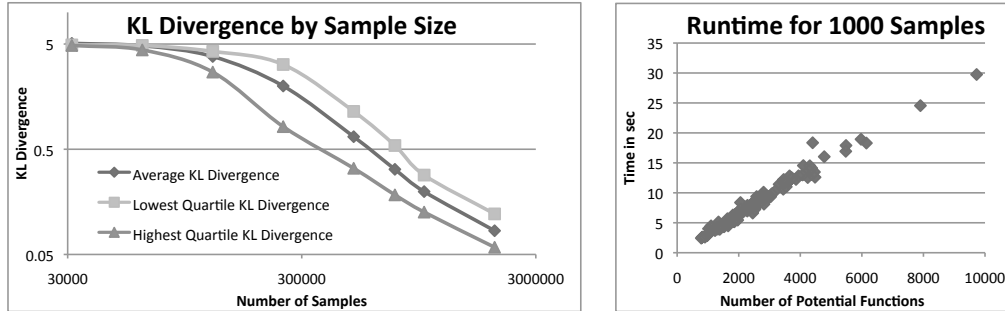

Figure 3: a) KL Divergence by sample size      b) Runtime for 1000 samples

prediction stage for each fold. We separate those values into two sets, $S_+, S_-$, based on whether the prediction turned out to be correct $(+)$ or incorrect $(-)$ when evaluated against the ground truth. Let $\sigma_+, \sigma_-$ denote the average standard deviation for those values in $S_+, S_-$ respectively. Our hypothesis is that we have higher confidence in the correct predictions, that is, $\sigma_+$ will typically be smaller than $\sigma_-$. In other words, we hypothesize that the relative difference between the average deviations, $\Delta(\sigma) = 2\frac{\sigma_- - \sigma_+}{\sigma_+ + \sigma_-}$, is larger than $0$. Under the corresponding null hypothesis, we would expect any difference in average standard deviation, and therefore any nonzero $\Delta(\sigma)$, to be purely coincidental or noise. Assuming that such noise in the $\Delta(\sigma)$'s, which we computed for each fold, can be approximated by a Gaussian distribution with $0$ mean and unknown variance[5], we test the null hypothesis using a two tailed Z-test with the observed sample variance. The Z-test scores on the 4 differently sized splits are reported in Table 2 b) and allow us to reject the null hypothesis with very high confidence. Table 2 b) also lists $\Delta(\sigma)$ for each split averaged across the multiple folds and shows that $\sigma_-$ is about 40% larger than $\sigma_+$ on average.

## 4.2 Algorithm performance

In investigating the performance of the sampling algorithm we are mainly interested in two questions: 1) How many samples does it take to converge on the marginal density functions? and 2) What is the computational cost of sampling? To answer the first question, we collect independent samples of varying size from 31 thousand to 2 million and one reference sample with 3 million steps for all folds. For each of the former samples we compare the marginals thus obtained to the ones of the reference sample by measuring their KL divergence. To compute the KL divergence we discretize the density function using a histogram with 10 bins. The center line in Figure 3 a) shows the average KL divergence with respect to the sample size across all folds. To study the impact of dimensionality on convergence, we order the folds by the number of random variables $n$ and show the average KL divergence for the lowest and highest quartile which contains $174-224$ and $322-413$ random variables respectively. The plot is drawn in log-log scale and therefore suggests that each magnitude increase in sample size yields a magnitude improvement in KL divergence. To answer the second question, Figure 3 b) displays the time needed to generate 1000 samples with respect to the number of potential functions in the CCMRF. Computing the induced probability density function along the sampled line segment dominates the cost of each sampling step and the graph shows that this cost grows linearly with the number of potential functions.

## 5 Conclusion

We have presented a novel approximation scheme for computing marginal probabilities over constrained continuous MRFs based on recent results in computational geometry and discussed techniques to improve its efficiency. We introduced an effective sampling algorithm and verified its performance in an empirical evaluation. To our knowledge, this is the first study of the theoretical, practical, and empirical aspects of marginal computation in general constrained continuous MRFs. While our initial results are quite promising, there are still many further directions for research including improved scalability, applications to other probabilistic inference problems, and using the confidence values to improve the prediction accuracy.

## Acknowledgment

We thank Stanley Kok, Stephan Bach, and the anonymous reviewers for their helpful comments and suggestions. This material is based upon work supported by the National Science Foundation under Grant No. 0937094. Any opinions, findings, and conclusions or recommendations expressed in this material are those of the authors and do not necessarily reflect the views of the NSF.

## Footnotes

[1]We ignore equality constraints for now until the discussion of the algorithm in Section 3.4

[2]The $O^*$ notation ignores logarithmic and factors and dependence on other parameters like error bounds.

[3]We used $\theta = 2$ in our experiments.

[4] `http://en.wikipedia.org/wiki/Wikipedia:Featured_lists`, see [3] for more information on the dataset

[5]Even if the standard deviations in $S_+, S_-$ are not normally distributed, the central limit theorem postulates that their averages will eventually follow a normal distributions under independence assumptions.

## References

[1] E. B Sudderth. *Graphical models for visual object recognition and tracking*. Ph.D. thesis, Massachusetts Institute of Technology, 2006.

[2] L. Lovasz and S. Vempala. Hit-and-run from a corner. In *Proceedings of the thirty-sixth annual ACM symposium on Theory of computing*, pages 310–314, Chicago, IL, USA, 2004. ACM.

[3] M. Broecheler, L. Mihalkova, and L. Getoor. Probabilistic similarity logic. In *Conference on Uncertainty in Artificial Intelligence*, 2010.

[4] M. Richardson and P. Domingos. Markov logic networks. *Machine Learning*, 62(1):107–136, 2006.

[5] K. Kersting and L. De Raedt. Bayesian logic programs. Technical report, Albert-Ludwigs University, 2001.

[6] B. Taskar, P. Abbeel, and D. Koller. Discriminative probabilistic models for relational data. In *Proceedings of UAI-02*, 2002.

[7] M. Broecheler, G. Simari, and VS. Subrahmanian. Using histograms to better answer queries to probabilistic logic programs. *Logic Programming*, page 4054, 2009.

[8] M. E. Dyer and A. M. Frieze. On the complexity of computing the volume of a polyhedron. *SIAM Journal on Computing*, 17(5):967–974, October 1988.

[9] R. Kannan, L. Lovasz, and M. Simonovits. Random walks and an o*(n5) volume algorithm for convex bodies. *Random structures and algorithms*, 11(1):150, 1997.

[10] R. L. Smith. Efficient monte carlo procedures for generating points uniformly distributed over bounded regions. *Operations Research*, 32(6):1296–1308, 1984.

[11] S. Agmon. The relaxation method for linear inequalities. *Canadian Journal of Mathematics*, 6(3):382392, 1954.

[12] T. S. Motzkin and I. J. Schoenberg. The relaxation method for linear inequalities. *IJ Schoenberg: Selected Papers*, page 75, 1988.

[13] P. Sen, G. Namata, M. Bilgic, L. Getoor, B. Galligher, and T. Eliassi-Rad. Collective classification in network data. *AI Magazine*, 29(3):93, 2008.

